# Adaptive optimal training of animal behavior

**Ji Hyun Bak**[1,4]  **Jung Yoon Choi**[2,3]  **Athena Akrami**[3,5]  **Ilana Witten**[2,3]  **Jonathan W. Pillow**[2,3]

[1]Department of Physics, [2]Department of Psychology, Princeton University
[3]Princeton Neuroscience Institute, Princeton University
[4]School of Computational Sciences, Korea Institute for Advanced Study
[5]Howard Hughes Medical Institute
jhbak@kias.re.kr, {jungchoi,aakrami,iwitten,pillow}@princeton.edu

## Abstract

Neuroscience experiments often require training animals to perform tasks designed to elicit various sensory, cognitive, and motor behaviors. Training typically involves a series of gradual adjustments of stimulus conditions and rewards in order to bring about learning. However, training protocols are usually hand-designed, relying on a combination of intuition, guesswork, and trial-and-error, and often require weeks or months to achieve a desired level of task performance. Here we combine ideas from reinforcement learning and adaptive optimal experimental design to formulate methods for adaptive optimal training of animal behavior. Our work addresses two intriguing problems at once: first, it seeks to infer the learning rules underlying an animal's behavioral changes during training; second, it seeks to exploit these rules to select stimuli that will maximize the rate of learning toward a desired objective. We develop and test these methods using data collected from rats during training on a two-interval sensory discrimination task. We show that we can accurately infer the parameters of a policy-gradient-based learning algorithm that describes how the animal's internal model of the task evolves over the course of training. We then formulate a theory for optimal training, which involves selecting sequences of stimuli that will drive the animal's internal policy toward a desired location in the parameter space. Simulations show that our method can in theory provide a substantial speedup over standard training methods. We feel these results will hold considerable theoretical and practical implications both for researchers in reinforcement learning and for experimentalists seeking to train animals.

## 1 Introduction

An important first step in many neuroscience experiments is to train animals to perform a particular sensory, cognitive, or motor task. In many cases this training process is slow (requiring weeks to months) or difficult (resulting in animals that do not successfully learn the task). This increases the cost of research and the time taken for experiments to begin, and poorly trained animals—for example, animals that incorrectly base their decisions on trial history instead of the current stimulus—may introduce variability in experimental outcomes, reducing interpretability and increasing the risk of false conclusions.

In this paper, we present a principled theory for the design of normatively optimal adaptive training methods. The core innovation is a synthesis of ideas from reinforcement learning and adaptive experimental design: we seek to reverse engineer an animal's internal learning rule from its observed behavior in order to select stimuli that will drive learning as quickly as possible toward a desired objective. Our approach involves estimating a model of the animal's internal state as it evolves over training sessions, including both the current policy governing behavior and the learning rule used to

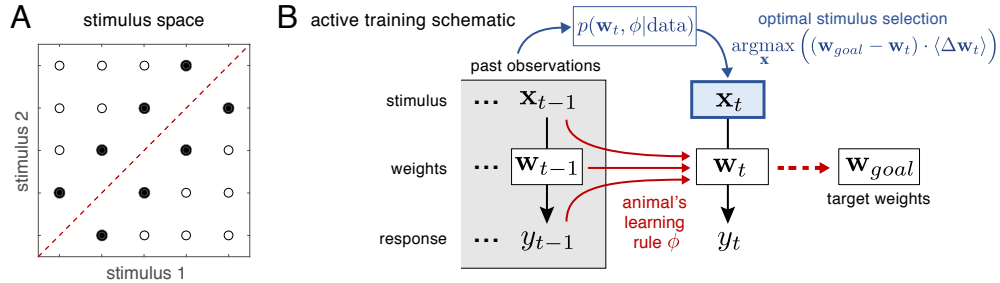

Figure 1: (A) Stimulus space for a 2AFC discrimination task, with optimal separatrix between correct "left" and "right" choices shown in red. Filled circles indicate a "reduced" set of stimuli (consisting of those closest to the decision boundary) which have been used in several prominent studies [3, 6, 9]. (B) Schematic of active training paradigm. We infer the animal's current weights $\mathbf{w}_t$ and its learning rule ("RewardMax"), parametrized by $\phi$, and use them to determine an optimal stimulus $\mathbf{x}_t$ for the current trial ("AlignMax"), where optimality is determined by the expected weight change towards the target weights $\mathbf{w}_{\text{goal}}$.

modify this policy in response to feedback. We model the animal as using a policy-gradient based learning rule [15], and show that parameters of this learning model can be successfully inferred from a behavioral time series dataset collected during the early stages of training. We then use the inferred learning rule to compute an optimal sequence of stimuli, selected adaptively on a trial-by-trial basis, that will drive the animal's internal model toward a desired state. Intuitively, optimal training involves selecting stimuli that maximally align the predicted change in model parameters with the trained behavioral goal, which is defined as a point in the space of model parameters. We expect this research to provide both practical and theoretical benefits: the adaptive optimal training protocol promises a significantly reduced training time required to achieve a desired level of performance, while providing new scientific insights into how and what animals learn over the course of the training period.

## 2 Modeling animal decision-making behavior

Let us begin by defining the ingredients of a generic decision-making task. In each trial, the animal is presented with a stimulus $\mathbf{x}$ from a bounded stimulus space $X$, and is required to make a choice $y$ among a finite set of available responses $Y$. There is a fixed reward map $r : \{X, Y\} \to \mathbb{R}$. It is assumed that this behavior is governed by some internal model, or the psychometric function, described by a set of parameters or weights $\mathbf{w}$. We introduce the "$y$-bar" notation $\bar{y}(\mathbf{x})$ to indicate the correct choice for the given stimulus $\mathbf{x}$, and let $X_y$ denote the "stimulus group" for a given $y$, defined as the set of all stimuli $\mathbf{x}$ that map to the same correct choice $y = \bar{y}(\mathbf{x})$.

For concreteness, we consider a two-alternative forced-choice (2AFC) discrimination task where the stimulus vector for each trial, $\mathbf{x} = (x_1, x_2)$, consists of a pair of scalar-valued stimuli that are to be compared [6, 8, 9, 16]. The animal should report either $x_1 > x_2$ or $x_1 < x_2$, indicating its choice with a left ($y = L$) or right ($y = R$) movement, respectively. This results in a binary response space, $Y = \{L, R\}$. We define the reward function $r(\mathbf{x}, y)$ to be a Boolean function that indicates whether a stimulus-response pair corresponds to a correct choice (which should therefore be rewarded) or not:

$$r(\mathbf{x}, y) = \begin{cases} 1 & \text{if } \{x_1 > x_2, \ y = L\} \text{ or } \{x_1 < x_2, \ y = R\}; \\ 0 & \text{otherwise.} \end{cases} \tag{1}$$

Figure 1A shows an example 2-dimensional stimulus space for such a task, with circles representing a discretized set of possible stimuli $X$, and the desired separatrix (the boundary separating the two stimulus groups $X_L$ and $X_R$) shown in red. In some settings, the experimenter may wish to focus on some "reduced" set of stimuli, as indicated here by filled symbols [3, 6, 9].

We model the animal's choice behavior as arising from a Bernoulli generalized linear model (GLM), also known as the logistic regression model. The choice probabilities for the two possible stimuli at trial $t$ are given by

$$p_R(\mathbf{x}_t, \mathbf{w}_t) = \frac{1}{1 + \exp(-\mathbf{g}(\mathbf{x}_t)^\top \mathbf{w}_t)}, \qquad p_L(\mathbf{x}_t, \mathbf{w}_t) = 1 - p_R(\mathbf{x}_t, \mathbf{w}_t) \tag{2}$$

where $\mathbf{g}(\mathbf{x}) = (1, \mathbf{x}^\top)^\top$ is the input carrier vector, and $\mathbf{w} = (b, \mathbf{a}^\top)^\top$ is the vector of parameters or weights governing behavior. Here $b$ describes the animal's internal bias to choosing "right" ($y = R$), and $\mathbf{a} = (a_1, a_2)$ captures the animal's sensitivity to the stimulus.[1]

We may also incorporate the trial history as additional dimensions of the input governing the animal's behavior; humans and animals alike are known to exhibit history-dependent behavior in trial-based tasks [1, 3, 5, 7]. Based on some preliminary observations from animal behavior (see Supplementary Material for details), we encode the trial history as a compressed stimulus history, using a binary variable $\epsilon_{\bar{y}(\mathbf{x})}$ defined as $\epsilon_L = -1$ and $\epsilon_R = +1$. Taking into account the previous $d$ trials, the input carrier vector and the weight vector become:

$$\mathbf{g}(\mathbf{x}_t) \to (1, \mathbf{x}_t^\top, \epsilon_{\bar{y}(\mathbf{x}_{t-1})}, \cdots, \epsilon_{\bar{y}(\mathbf{x}_{t-d})})^\top, \qquad \mathbf{w}_t \to (b, \mathbf{a}^\top, h_1, \cdots, h_d). \tag{3}$$

The history dependence parameter $h_d$ describes the animal's tendency to stick to the correct answer from the previous trial ($d$ trials back). Because varying number of history terms $d$ gives a family of psychometric models, determining the optimal $d$ is a well-defined model selection problem.

## 3  Estimating time-varying psychometric function

In order to drive the animal's performance toward a desired objective, we first need a framework to describe, and accurately estimate, the *time-varying* model parameters of the animal behavior, which is fundamentally non-stationary while training is in progress.

### 3.1  Constructing the random walk prior

We assume that the single-step weight change at each trial $t$ follows a random walk, $w_t - w_{t-1} = \xi_t$, where $\xi_t \sim \mathcal{N}(0, \sigma_t^2)$, for $t = 1, \cdots, N$. Let $w_0$ be some prior mean for the initial weight. We assume $\sigma_2 = \cdots = \sigma_N = \sigma$, which is to believe that although the behavior is variable, the *variability* of the behavior is a constant property of the animal. We can write this more concisely using a state-space representation [2, 11], in terms of the vector of time-varying weights $\mathbf{w} = (w_1, w_2, \cdots, w_N)^\top$ and its prior mean $\mathbf{w}_0 = w_0 \mathbf{1}$:

$$D(\mathbf{w} - \mathbf{w}_0) = \boldsymbol{\xi} \sim \mathcal{N}(\mathbf{0}, \Sigma), \tag{4}$$

where $\Sigma = \mathrm{diag}(\sigma_1^2, \sigma^2, \cdots, \sigma^2)$ is the $N \times N$ covariance matrix, and $D$ is the sparse banded matrix with first row of an identity matrix and subsequent rows computing first order differences. Rearranging, the full random walk prior on the $N$-dimensional vector $\mathbf{w}$ is

$$\mathbf{w} \sim \mathcal{N}(\mathbf{w}_0, C), \quad \text{where} \quad C^{-1} = D^\top \Sigma^{-1} D. \tag{5}$$

In many practical cases there are multiple weights in the model, say $K$ weights. The full set of parameters should now be arranged into an $N \times K$ array of weights $\{w_{ti}\}$, where the two subscripts consistently indicate the trial number ($t = 1, \cdots, N$) and the type of parameter ($i = 1, \cdots, K$), respectively. This gives a matrix

$$W = \{w_{ti}\} = (\mathbf{w}_{*1}, \cdots, \mathbf{w}_{*i}, \cdots, \mathbf{w}_{*K}) = (\mathbf{w}_{1*}, \cdots, \mathbf{w}_{t*}, \cdots, \mathbf{w}_{N*})^\top \tag{6}$$

where we denote the vector of all weights at trial $t$ as $\mathbf{w}_{t*} = (w_{t1}, w_{t2}, \cdots, w_{tK})^\top$, and the time series of the $i$-th weight as $\mathbf{w}_{*i} = (w_{1i}, w_{2i}, \cdots, w_{Ni})^\top$.

Let $\mathbf{w} = \mathrm{vec}(W) = (\mathbf{w}_{*1}^\top, \cdots, \mathbf{w}_{*K}^\top)^\top$ be the vectorization of $W$, a long vector with the columns of $W$ stacked together. Equation (5) still holds for this extended weight vector $\mathbf{w}$, where the extended $D$ and $\Sigma$ are written as block diagonal matrices $D = \mathrm{diag}(D_1, D_2, \cdots, D_K)$ and $\Sigma = \mathrm{diag}(\Sigma_1, \Sigma_2, \cdots, \Sigma_K)$, respectively, where $D_i$ is the weight-specific $N \times N$ difference matrix and $\Sigma_i$ is the corresponding covariance matrix. Within a linear model one can freely renormalize the units of the stimulus space in order to keep the sizes of all weights comparable, and keep all $\Sigma_i$'s equal. We used a transformed stimulus space in which the center is at 0 and the standard deviation is 1.

## 3.2  Log likelihood

Let us denote the log likelihood of the observed data by $L = \sum_{t=1}^{N} L_t$, where $L_t = \log p(y_t | \mathbf{x}_t, \mathbf{w}_{t*})$ is the trial-specific log likelihood. Within the binomial model we have

$$L_t = (1 - \delta_{y_t, R}) \log(1 - p_R(\mathbf{x}_t, \mathbf{w}_{t*})) + \delta_{y_t, R} \log p_R(\mathbf{x}_t, \mathbf{w}_{t*}). \tag{7}$$

Abbreviating $p_R(\mathbf{x}_t, \mathbf{w}_{t*}) = p_t$ and $p_L(\mathbf{x}_t, \mathbf{w}_{t*}) = 1 - p_t$, the trial-specific derivatives are solved to be $\partial L_t / \partial \mathbf{w}_{t*} = (\delta_{y_t, R} - p_t)\, \mathbf{g}(\mathbf{x}_t) \equiv \mathbf{\Delta}_t$ and $\partial^2 L_t / \partial \mathbf{w}_{t*} \partial \mathbf{w}_{t*} = -p_t(1 - p_t)\mathbf{g}(\mathbf{x}_t)\mathbf{g}(\mathbf{x}_t)^{\top} \equiv \Lambda_t$. Extension to the full weight vector is straightforward because distinct trials do not interact. Working out with the indices, we may write

$$\frac{\partial L}{\partial \mathbf{w}} = \mathrm{vec}([\mathbf{\Delta}_1, \cdots, \mathbf{\Delta}_N]^{\top}), \qquad \frac{\partial^2 L}{\partial \mathbf{w}^2} = \begin{bmatrix} M_{11} & M_{12} & \cdots & M_{1K} \\ M_{21} & M_{22} & & M_{2K} \\ \vdots & & \ddots & \vdots \\ M_{K1} & M_{K2} & \cdots & M_{KK} \end{bmatrix} \tag{8}$$

where the $(i, j)$-th block of the full second derivative matrix is an $N \times N$ diagonal matrix defined by $M_{ij} = \partial^2 L / \partial \mathbf{w}_{*i} \partial \mathbf{w}_{*j} = \mathrm{diag}((\Lambda_1)_{ij}, \cdots, (\Lambda_t)_{ij}, \cdots, (\Lambda_N)_{ij})$. After this point, we can simplify our notation such that $\mathbf{w}_t = \mathbf{w}_{t*}$. The weight-type-specific $\mathbf{w}_{*i}$ will no longer appear.

## 3.3  MAP estimate of $\mathbf{w}$

The posterior distribution of $\mathbf{w}$ is a combination of the prior and the likelihood (Bayes' rule):

$$\log p(\mathbf{w} | \mathcal{D}) \sim \left( \frac{1}{2} \log |C^{-1}| - \frac{1}{2}(\mathbf{w} - \mathbf{w}_0)^{\top} C^{-1}(\mathbf{w} - \mathbf{w}_0) \right) + L. \tag{9}$$

We can perform a numerical maximization of the log posterior using Newton's method (we used the Matlab function `fminunc`), knowing its gradient $\mathbf{j}$ and the hessian $H$ explicitly:

$$\mathbf{j} = \frac{\partial(\log p)}{\partial \mathbf{w}} = -C^{-1}(\mathbf{w} - \mathbf{w}_0) + \frac{\partial L}{\partial \mathbf{w}}, \qquad H = \frac{\partial^2(\log p)}{\partial \mathbf{w}^2} = -C^{-1} + \frac{\partial^2 L}{\partial \mathbf{w}^2}. \tag{10}$$

The maximum a posteriori (MAP) estimate $\hat{\mathbf{w}}$ is where the gradient vanishes, $\mathbf{j}(\hat{\mathbf{w}}) = \mathbf{0}$. If we work with a Laplace approximation, the posterior covariance is $\mathrm{Cov} = -\mathrm{H}^{-1}$ evaluated at $\mathbf{w} = \hat{\mathbf{w}}$.

## 3.4  Hyperparameter optimization

The model hyperparameters consist of $\sigma_1$, governing the variance of $w_1$, the weights on the first trial of a session, and $\sigma$, governing the variance of the trial-to-trial diffusive change of the weights. To set these hyperparameters, we fixed $\sigma_1$ to a large default value, and used maximum marginal likelihood or "evidence optimization" over a fixed grid of $\sigma$ [4, 11, 13]. The marginal likelihood is given by:

$$p(\mathbf{y} | \mathbf{x}, \sigma) = \int d\mathbf{w}\, p(\mathbf{y} | \mathbf{x}, \mathbf{w}) p(\mathbf{w} | \sigma) = \frac{p(\mathbf{y} | \mathbf{x}, \mathbf{w}) p(\mathbf{w} | \sigma)}{\mathbf{p}(\mathbf{w} | \mathbf{x}, \mathbf{y}, \sigma)} \approx \frac{\exp(L) \cdot \mathcal{N}(\mathbf{w} | \mathbf{w}_0, C)}{\mathcal{N}(\mathbf{w} | \hat{\mathbf{w}}, -H^{-1})}, \tag{11}$$

where $\hat{\mathbf{w}}$ is the MAP estimate of the entire vector of time-varying weights and $H$ is the Hessian of the log-posterior over $\mathbf{w}$ at its mode. This formula for marginal likelihood results from the well-known Laplace approximation to the posterior [11, 12]. We found the estimate not to be insensitive to $\sigma_1$ so long as it is sufficiently large.

## 3.5  Application

We tested our method using a simulation, drawing binary responses from a stimulus-free GLM $y_t \sim \mathrm{logistic}(w_t)$, where $w_t$ was diffused as $w_{t+1} \sim \mathcal{N}(w_t, \sigma^2)$ with a fixed hyperparameter $\sigma$. Given the time series of responses $\{y_t\}$, our method captures the true $\sigma$ through evidence maximization, and provides a good estimate of the time-varying $\mathbf{w} = \{w_t\}$ (Figure 2A). Whereas the estimate of the weight $w_t$ is robust over independent realizations of the responses, the instantaneous weight changes $\Delta w = w_{t+1} - w_t$ are not reproducible across realizations (Figure 2B). Therefore it is difficult to analyze the trial-to-trial weight changes directly from real data, where only one realization of the learning process is accessible.

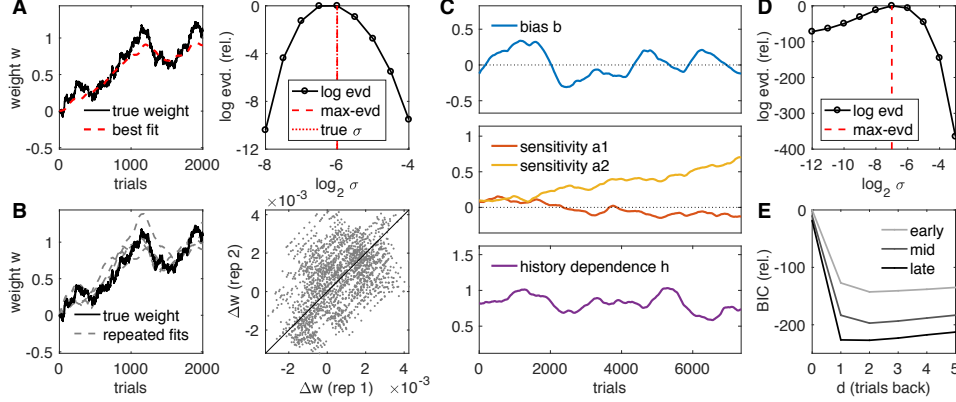

Figure 2: Estimating time-varying model parameters. (A-B) Simulation: (A) Our method captures the true underlying variability $\sigma$ by maximizing evidence. (B) Weight estimates are accurate and robust over independent realizations of the responses, but weight changes across realizations are not reproducible. (C-E) From the choice behavior of a rat under training, we could (C) estimate the time-varying weights of its psychometric model, and (D) determine the characteristic variability by evidence maximization. (E) The number of history terms to be included in the model was determined by comparing the BIC, using the early/mid/late parts of the rat dataset. Because log-likelihood is calculated up to a constant normalization, both log-evidence and BIC are shown in relative values.

We also applied our method to an actual experimental dataset from rats during the early training period for a 2AFC discrimination task, as introduced in Section 2 (using classical training methods [3], see Supplementary Material for detailed description). We estimated the time-varying weights of the GLM (Figure 2C), and estimated the characteristic variability of the rat behavior $\sigma_{\mathrm{rat}} = 2^{-7}$ by maximizing marginal likelihood (Figure 2D). To determine the length $d$ of the trial history dependence, we fit models with varying $d$ and used the Bayesian Information Criterion $\mathrm{BIC}(d) = -2 \log L(d) + K(d) \log N$ (Figure 2E). We found that animal behavior exhibits long-range history depedence at the beginning of training, but this dependence becomes shorter as training progresses. Near the end of the dataset, the behavior of the rat is best described $d_{\mathrm{rat}} = 1$ (single-trial history dependence), and we use this value for the remainder of our analyses.

## 4   Incorporating learning

The fact that animals show improved performance, as training progresses, suggests that we need a non-random component in our model that accounts for learning. We will first introduce a simple model of weight change based on the ideas from reinforcement learning, then discuss how we can incorporate the learning model into our time-varying estimate method.

A good candidate model for animal learning is the policy gradient update from reinforcement learning, for example as in [15]. There are debates as to whether animals actually learn using policy-based methods, but it is difficult to define a reasonable value function that is consistent with our preliminary observations of rat behavior (e.g. win-stay/lose-switch). A recent experimental study supports the use of policy-based models in human learning behavior [10].

### 4.1   RewardMax model of learning (policy gradient update)

Here we consider a simple model of learning, in which the learner attempts to update its policy (here the weight parameters in the model) to maximize the expected reward. Given some fixed reward function $r(\mathbf{x}, y)$, the expected reward at the next-upcoming trial $t$ is defined as

$$\rho(\mathbf{w}_t) = \left\langle \left\langle r(\mathbf{x}_t, y_t) \right\rangle_{p(y_t|\mathbf{x}_t, \mathbf{w}_t)} \right\rangle_{P_X(\mathbf{x}_t)} \tag{12}$$

where $P_X(\mathbf{x}_t)$ reflects the subject animal's knowledge as to the probability that a given stimulus $\mathbf{x}$ will be presented at trial $t$, which may be dynamically updated. One way to construct the empirical

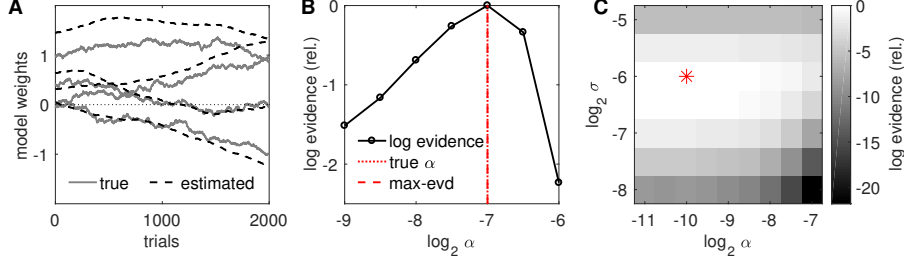

Figure 3: Estimating the learning model. (A-B) Simulated learner with $\sigma_{\mathrm{sim}} = \alpha_{\mathrm{sim}} = 2^{-7}$. (A) The four weight parameters of the simulated model are successfully recovered by our MAP estimate with the learning effect incorporated, where (B) the learning rate $\alpha$ is accurately determined by evidence maximization. (C) Evidence maximization analysis on the rat training dataset reveals $\sigma_{\mathrm{rat}} = 2^{-6}$ and $\alpha_{\mathrm{rat}} = 2^{-10}$. Displayed is a color plot of log evidence on the hyperparameter plane (in relative values). The optimal set of hyperparameters is marked with a star.

$P_X$ is to accumulate the stimulus statistics up to some timescale $\tau \geq 0$; here we restrict to the simplest limit $\tau = 0$, where only the most recent stimulus is remembered. That is, $P_X(\mathbf{x}_t) = \delta(\mathbf{x}_t - \mathbf{x}_{t-1})$. In practice $\rho$ can be evaluated at $\mathbf{w}_t = \mathbf{w}_{t-1}$, the posterior mean from previous observations.

Under the GLM (2), the choice probability is $p(y|\mathbf{x}, \mathbf{w}) = 1/(1 + \exp(-\epsilon_y \mathbf{g}(\mathbf{x})^\top \mathbf{w}))$, where $\epsilon_L = -1$ and $\epsilon_R = +1$, trial index suppressed. Therefore the expected reward can be written out explicitly, as well as its gradient with respect to $\mathbf{w}$:

$$\frac{\partial \rho}{\partial \mathbf{w}} = \sum_{\mathbf{x} \in X} P_X(\mathbf{x})\, f(\mathbf{x})\, p_R(\mathbf{x}, \mathbf{w})\, p_L(\mathbf{x}, \mathbf{w})\, \mathbf{g}(\mathbf{x}) \tag{13}$$

where we define the effective reward function $f(\mathbf{x}) \equiv \sum_{y \in Y} \epsilon_y r(\mathbf{x}, y)$ for each stimulus. In the spirit of the policy gradient update, we consider the **RewardMax model of learning**, which assumes that the animal will try to climb up the gradient of the expected reward by

$$\Delta \mathbf{w}_t = \alpha \left. \frac{\partial \rho}{\partial \mathbf{w}} \right|_t \equiv \mathbf{v}(\mathbf{w}_t, \mathbf{x}_t; \phi), \tag{14}$$

where $\Delta \mathbf{w}_t = (\mathbf{w}_{t+1} - \mathbf{w}_t)$. In this simplest setting, the learning rate $\alpha$ is the only learning hyperparameter $\phi = \{\alpha\}$. The model can be extended by incorporating more realistic aspects of learning, such as the non-isotropic learning rate, the rate of weight decay (forgetting), or the skewness between experienced and unexperienced rewards. For more discussion, see Supplementary Material.

## 4.2 Random walk prior with drift

Because our observation of a given learning process is stochastic and the estimate of the weight change is not robust (Figure 2B), it is difficult to test the learning rule (14) on any individual dataset. However, we can still assume that the learning rule underlies the observed weight changes as

$$\langle \Delta \mathbf{w} \rangle = \mathbf{v}(\mathbf{w}, \mathbf{x}; \phi) \tag{15}$$

where the average $\langle \cdot \rangle$ is over hypothetical repetitions of the same learning process. This effect of non-random learning can be incorporated into our random walk prior as a drift term, to make a fully Bayesian model for an imperfect learner. The new weight update prior is written as $D(\mathbf{w} - \mathbf{w}_0) = \mathbf{v} + \boldsymbol{\xi}$, where $\mathbf{v}$ is the "drift velocity" and $\boldsymbol{\xi} \sim \mathcal{N}(\mathbf{0}, \Sigma)$ is the noise. The modified prior is

$$\mathbf{w} - D^{-1}\mathbf{v} \sim \mathcal{N}(\mathbf{w}_0, C), \qquad C^{-1} = D^\top \Sigma^{-1} D. \tag{16}$$

Equations (9-10) can be re-written with the additional term $D^{-1}\mathbf{v}$. For the RewardMax model $\mathbf{v} = \alpha \partial \rho / \partial \mathbf{w}$, in particular, the first and second derivatives of the modified log posterior can be written out analytically. Details can be found in Supplementary Material.

## 4.3 Application

To test the model with drift, a simulated RewardMax learner was generated, based on the same task structure as in the rat experiment. The two hyperparameters $\{\sigma_{\mathrm{sim}}, \alpha_{\mathrm{sim}}\}$ were chosen such that the

resulting time series data is qualitatively similar to the rat data. The simulated learning model can be recovered by maximizing the evidence (11), now with the learning hyperparameter $\alpha$ as well as the variability $\sigma$. The solution accurately reflects the true $\alpha_{\text{sim}}$, shown where $\sigma$ is fixed at the true $\sigma_{\text{sim}}$ (Figures 3A-3B). Likewise, the learning model of a real rat was obtained by performing a grid search on the full hyperparameter plane $\{\sigma, \alpha\}$. We get $\sigma_{\text{rat}} = 2^{-6}$ and $\alpha_{\text{rat}} = 2^{-10}$ (Figure 3C). [2]

Can we determine whether the rat's behavior is in a regime where the effect of learning dominates the effect of noise, or vice versa? The obtained values of $\sigma$ and $\alpha$ depend on our choice of units which is arbitrary; more precisely, $\alpha \sim [\mathbf{w}^2]$ and $\sigma \sim [\mathbf{w}]$ where $[\mathbf{w}]$ scales as the weight. Dimensional analysis suggests a (dimensionless) order parameter $\beta = \alpha/\sigma^2$, where $\beta \gg 1$ would indicate a regime where the effect of learning is larger than the effect of noise. Our estimate of the hyperparameters gives $\beta_{\text{rat}} = \alpha_{\text{rat}}/\sigma_{\text{rat}}^2 \approx 4$, which leaves us optimistic.

## 5  AlignMax: Adaptive optimal training

Whereas the goal of the learner/trainee is (presumably) to maximize the expected reward, the trainer's goal is to drive the behavior of the trainee as close as possible to some fixed model that corresponds to a desirable, yet hypothetically achievable, performance. Here we propose a simple algorithm that aims to align the expected model parameter change of the trainee $\langle \Delta \mathbf{w}_t \rangle = \mathbf{v}(\mathbf{w}_t, \mathbf{x}_t; \phi)$ towards a fixed goal $\mathbf{w}_{\text{goal}}$. We can summarize this in an **AlignMax training formula**

$$\mathbf{x}_{t+1} = \underset{\mathbf{x}}{\text{argmax}} \ (\mathbf{w}_{\text{goal}} - \mathbf{w}_t)^\top \langle \Delta \mathbf{w}_t \rangle. \tag{17}$$

Looking at Equations (13), (14) and (17), it is worth noting that $\mathbf{g}(\mathbf{x})$ puts a heavier weight on more distinguishable or "easier" stimuli (exploitation), while $p_L p_R$ puts more weight on more difficult stimuli, with more uncertainty (exploration); an exploitation-exploration tradeoff emerges naturally.

We tested the AlignMax training protocol[3] using a simulated learner with fixed hyperparameters $\alpha_{\text{sim}} = 0.005$ and $\sigma_{\text{sim}} = 0$, using $\mathbf{w}_{\text{goal}} = (b, a_1, a_2, h)_{\text{goal}} = (0, -10, 10, 0)$ in the current paradigm. We chose a noise-free learner for clear visualization, but the algorithm works as well in the presence of noise ($\sigma > 0$, see Supplementary Material for a simulated noisy learner). As expected, our AlignMax algorithm achieves a much faster training compared to the usual algorithm where stimuli are presented randomly (Figure 4). The task performance was measured in terms of the success rate, the expected reward (12), and the Kullback-Leibler (KL) divergence. The KL divergence is defined as $D_{KL} = \sum_{\mathbf{x} \in X} P_X(\mathbf{x}) \sum_{y \in Y} \hat{p}_y(\mathbf{x}) \log(\hat{p}_y(\mathbf{x})/p_y(\mathbf{x}))$ where $\hat{p}_y(\mathbf{x}) = r(\mathbf{x}, y)$ is the "correct" psychometric function, and a smaller value of $D_{KL}$ indicates a behavior that is closer to the ideal. Both the expected reward and the KL divergence were evaluated using a uniform stimulus distribution $P_X(\mathbf{x})$. The low success rate is a distinctive feature of the adaptive training algorithm, which selects adversarial stimuli such that the "lazy flukes" are actively prevented (e.g. such that a left-biased learner wouldn't get thoughtless rewards from the left side). It is notable that the AlignMax training eliminates the bias $b$ and the history dependence $h$ (the two stimulus-independent parameters) much more quickly compared to the conventional (random) algorithm, as shown in Figure 4A.

Two general rules were observed from the optimal trainer. First, while the history dependence $h$ is non-zero, AlignMax alternates between different stimulus groups in order to suppress the win-stay behavior; once $h$ vanishes, AlignMax tries to neutralize the bias $b$ by presenting more stimuli from the "non-preferred" stimulus group yet being careful not to re-install the history dependence. For example, it would give $LLRLLR...$ for an $R$-biased trainee. This suggests that a pre-defined, non-adaptive de-biasing algorithm may be problematic as it may reinforce an unwanted history dependence (see Supp. Figure S1). Second, AlignMax exploits the full stimulus space by starting from some "easier" stimuli in the early stage of training (farther away from the true separatrix $x_1 = x_2$), and presenting progressively more difficult stimuli (closer to the separatrix) as the trainee performance improves. This suggests that using the reduced stimulus space may be suboptimal for training purposes. Indeed, training was faster on the full stimulus plane, than on the reduced set (Figures 4B-4C).

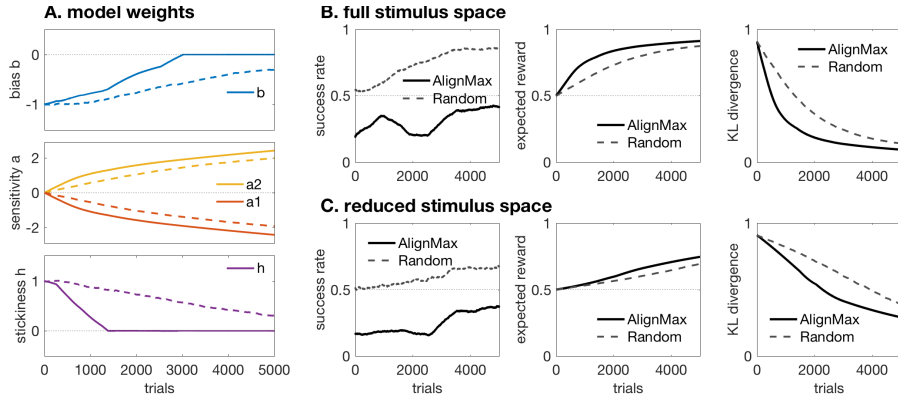

Figure 4: AlignMax training (solid lines) compared to a random training (dashed lines), for a simulated noise-free learner. (A) Weights evolving as training progresses, shown from a simulated training on the full stimulus space shown in Figure 1A. (B-C) Performances measured in terms of the success rate (moving average over 500 trials), the expected reward and the KL divergence. The simulated learner was trained either (B) in the full stimulus space, or (C) in the reduced stimulus space. The low success rate is a natural consequence of the active training algorithm, which tends to select adversarial stimuli to facilitate learning.

# 6 Discussion

In this work, we have formulated a theory for designing an optimal training protocol of animal behavior, which works adaptively to drive the current internal model of the animal toward a desired, pre-defined objective state. To this end, we have first developed a method to accurately estimate the time-varying parameters of the psychometric model directly from animal's behavioral time series, while characterizing the intrinsic variability $\sigma$ and the learning rate $\alpha$ of the animal by empirical Bayes. Interestingly, a dimensional analysis based on our estimate of the learning model suggests that the rat indeed lives in a regime where the effect of learning is stronger than the effect of noise.

Our method to infer the learning model from data is different from many conventional approaches of inverse reinforcement learning, which also seek to infer the underlying learning rules from externally observable behavior, but usually rely on the stationarity of the policy or the value function. On the contrary, our method works directly on the non-stationary behavior. Our technical contribution is twofold: first, building on the existing framework for estimation of state-space vectors [2, 11, 14], we provide a case in which parameters of a non-stationary model are successfully inferred from real time-series data; second, we develop a natural extension of the existing Bayesian framework where non-random model change (learning) is incorporated into the prior information.

The AlignMax optimal trainer provides important insights into the general principles of effective training, including a balanced strategy to neutralize both the bias and the history dependence of the animal, and a dynamic tradeoff between difficult and easy stimuli that makes efficient use of a broad range of the stimulus space. There are, however, two potential issues that may be detrimental to the practical success of the algorithm: First, the animal may suffer a loss of motivation due to the low success rate, which is a natural consequence of the adaptive training algorithm. Second, as with any model-based approach, mismatch of either the psychometric model (logistic, or any generalization model) or the learning model (RewardMax) may result in poor performances of the training algorithm. These issues are subject to tests on real training experiments. Otherwise, the algorithm is readily applicable. We expect it to provide both a significant reduction in training time and a set of reliable measures to evaluate the training progress, powered by direct access to the internal learning model of the animal.

### Acknowledgments

JHB was supported by the Samsung Scholarship and the NSF PoLS program. JWP was supported by grants from the McKnight Foundation, Simons Collaboration on the Global Brain (SCGB AWD1004351) and the NSF CAREER Award (IIS-1150186). We thank Nicholas Roy for the careful reading of the manuscript.

## Footnotes

[1] We use a convention in which a single-indexed tensor object is automatically represented as a column vector (in boldface notation), and the operation $(\cdot, \cdot, \cdots)$ concatenates objects horizontally.

[2] Based on a 2000-trial subset of the rat dataset.

[3] When implementing the algorithm within the current task paradigm, because of the way we model the history variable as part of the stimulus, it is important to allow the algorithm to choose up to $d + 1$ future stimuli, in this case as a pair $\{\mathbf{x}_{t+1}, \mathbf{x}_{t+2}\}$, in order to generate a desired pattern of trial history.

# References

[1] A. Abrahamyan, L. L. Silva, S. C. Dakin, M. Carandini, and J. L. Gardner. Adaptable history biases in human perceptual decisions. *Proc. Nat. Acad. Sci.*, 113(25):E3548–E3557, 2016.

[2] Y. Ahmadian, J. W. Pillow, and L. Paninski. Efficient Markov chain Monte Carlo methods for decoding neural spike trains. *Neural Computation*, 23(1):46–96, 2011.

[3] A. Akrami, C. Kopec, and C. Brody. Trial history vs. sensory memory - a causal study of the contribution of rat posterior parietal cortex (ppc) to history-dependent effects in working memory. *Society for Neuroscience Abstracts*, 2016.

[4] C. M. Bishop. *Pattern Recognition and Machine Learning*. Information science and statistics. Springer, 2006.

[5] L. Busse, A. Ayaz, N. T. Dhruv, S. Katzner, A. B. Saleem, M. L. Schölvinck, A. D. Zaharia, and M. Carandini. The detection of visual contrast in the behaving mouse. *J. Neurosci.*, 31(31):11351–11361, 2011.

[6] A. Fassihi, A. Akrami, V. Esmaeili, and M. E. Diamond. Tactile perception and working memory in rats and humans. *Proc. Nat. Acad. Sci.*, 111(6):2331–2336, 2014.

[7] I. Fründ, F. A. Wichmann, and J. H. Macke. Quantifying the effect of intertrial dependence on perceptual decisions. *J. Vision*, 14(7):9–9, 2014.

[8] D. M. Green and J. A. Swets. *Signal Detection Theory and Psychophysics*. Wiley, New York, 1966.

[9] A. Hernández, E. Salinas, R. García, and R. Romo. Discrimination in the sense of flutter: new psychophysical measurements in monkeys. *J. Neurosci.*, 17(16):6391–6400, 1997.

[10] J. Li and N. D. Daw. Signals in human striatum are appropriate for policy update rather than value prediction. *J. Neurosci.*, 31(14):5504–5511, 2011.

[11] L. Paninski, Y. Ahmadian, D. G. Ferreira, S. Koyama, K. Rahnama Rad, M. Vidne, J. Vogelstein, and W. Wu. A new look at state-space models for neural data. *J. Comp. Neurosci.*, 29(1):107–126, 2010.

[12] J. W. Pillow, Y. Ahmadian, and L. Paninski. Model-based decoding, information estimation, and change-point detection techniques for multineuron spike trains. *Neural Comput*, 23(1):1–45, Jan 2011.

[13] M. Sahani and J. F. Linden. Evidence optimization techniques for estimating stimulus-response functions. In S. Becker, S. Thrun, and K. Obermayer, editors, *Adv. Neur. Inf. Proc. Sys. 15*, pages 317–324. MIT Press, 2003.

[14] A. C. Smith, L. M. Frank, S. Wirth, M. Yanike, D. Hu, Y. Kubota, A. M. Graybiel, W. A. Suzuki, and E. N. Brown. Dynamic analysis of learning in behavioral experiments. *J. Neurosci.*, 24(2):447–461, 2004.

[15] R. S. Sutton, D. Mcallester, S. Singh, and Y. Mansour. Policy gradient methods for reinforcement learning with function approximation. In S. A. Solla, T. K. Leen, and K. Muller, editors, *Adv. Neur. Inf. Proc. Sys. 12*, pages 1057–1063. MIT Press, 2000.

[16] C. W. Tyler and C.-C. Chen. Signal detection theory in the 2afc paradigm: Attention, channel uncertainty and probability summation. *Vision Research*, 40(22):3121–3144, 2000.

